# Rethinking LDA: Why Priors Matter

**Hanna M. Wallach    David Mimno    Andrew McCallum**
Department of Computer Science
University of Massachusetts Amherst
Amherst, MA 01003
{wallach,mimno,mccallum}@cs.umass.edu

## Abstract

Implementations of topic models typically use symmetric Dirichlet priors with fixed concentration parameters, with the implicit assumption that such "smoothing parameters" have little practical effect. In this paper, we explore several classes of structured priors for topic models. We find that an asymmetric Dirichlet prior over the document–topic distributions has substantial advantages over a symmetric prior, while an asymmetric prior over the topic–word distributions provides no real benefit. Approximation of this prior structure through simple, efficient hyperparameter optimization steps is sufficient to achieve these performance gains. The prior structure we advocate substantially increases the robustness of topic models to variations in the number of topics and to the highly skewed word frequency distributions common in natural language. Since this prior structure can be implemented using efficient algorithms that add negligible cost beyond standard inference techniques, we recommend it as a new standard for topic modeling.

## 1    Introduction

Topic models such as latent Dirichlet allocation (LDA) [3] have been recognized as useful tools for analyzing large, unstructured collections of documents. There is a significant body of work applying LDA to an wide variety of tasks including analysis of news articles [14], study of the history of scientific ideas [2, 9], topic-based search interfaces[1] and navigation tools for digital libraries [12]. In practice, users of topic models are typically faced with two immediate problems: First, extremely common words tend to dominate all topics. Second, there is relatively little guidance available on how to set $T$, the number of topics, or studies regarding the effects of using a suboptimal setting for $T$. Standard practice is to remove "stop words" before modeling using a manually constructed, corpus-specific stop word list and to optimize $T$ by either analyzing probabilities of held-out documents or resorting to a more complicated nonparametric model. Additionally, there has been relatively little work in the machine learning literature on the structure of the prior distributions used in LDA: most researchers simply use symmetric Dirichlet priors with heuristically set concentration parameters. Asuncion et al. [1] recently advocated inferring the concentration parameters of these symmetric Dirichlets from data, but to date there has been no rigorous scientific study of the priors used in LDA—from the choice of prior (symmetric versus asymmetric Dirichlets) to the treatment of hyperparameters (optimize versus integrate out)—and the effects of these modeling choices on the probability of held-out documents and, more importantly, the quality of inferred topics. In this paper, we demonstrate that practical implementation issues (handling stop words, setting the number of topics) and theoretical issues involving the structure of Dirichlet priors are intimately related.

We start by exploring the effects of classes of hierarchically structured Dirichlet priors over the document–topic distributions and topic–word distributions in LDA. Using MCMC simulations, we find that using an asymmetric, hierarchical Dirichlet prior over the document–topic distributions and

a symmetric Dirichlet prior over the topic–word distributions results in significantly better model performance, measured both in terms of the probability of held-out documents and in the quality of inferred topics. Although this hierarchical Bayesian treatment of LDA produces good results, it is computationally intensive. We therefore demonstrate that optimizing the hyperparameters of asymmetric, nonhierarchical Dirichlets as part of an iterative inference algorithm results in similar performance to the full Bayesian model while adding negligible computational cost beyond standard inference techniques. Finally, we show that using optimized Dirichlet hyperparameters results in dramatically improved consistency in topic usage as $T$ is increased. By decreasing the sensitivity of the model to the number of topics, hyperparameter optimization results in robust, data-driven models with substantially less model complexity and computational cost than nonparametric models. Since the priors we advocate (an asymmetric Dirichlet over the document–topic distributions and a symmetric Dirichlet over the topic–word distributions) have significant modeling benefits and can be implemented using highly efficient algorithms, we recommend them as a new standard for LDA.

## 2 Latent Dirichlet Allocation

LDA is a generative topic model for documents $\mathcal{W} = \{\boldsymbol{w}^{(1)}, \boldsymbol{w}^{(2)}, \ldots, \boldsymbol{w}^{(D)}\}$. A "topic" $t$ is a discrete distribution over words with probability vector $\boldsymbol{\phi}_t$. A Dirichlet prior is placed over $\Phi = \{\boldsymbol{\phi}_1, \ldots \boldsymbol{\phi}_T\}$. In almost all previous work on LDA, this prior is assumed to be symmetric (i.e., the base measure is fixed to a uniform distribution over words) with concentration parameter $\beta$:

$$P(\Phi) = \prod_t \mathrm{Dir}\left(\boldsymbol{\phi}_t; \beta \boldsymbol{u}\right) = \prod_t \frac{\Gamma(\beta)}{\prod_w \Gamma(\frac{\beta}{W})} \prod_w \phi_{w|t}^{\frac{\beta}{W}-1} \delta\left(\sum_w \phi_{w|t} - 1\right). \qquad (1)$$

Each document, indexed by $d$, has a document-specific distribution over topics $\boldsymbol{\theta}_d$. The prior over $\Theta = \{\boldsymbol{\theta}_1, \ldots \boldsymbol{\theta}_D\}$ is also assumed to be a symmetric Dirichlet, this time with concentration parameter $\alpha$. The tokens in every document $\boldsymbol{w}^{(d)} = \{w_n^{(d)}\}_{n=1}^{N_d}$ are associated with corresponding topic assignments $\boldsymbol{z}^{(d)} = \{z_n^{(d)}\}_{n=1}^{N_d}$, drawn i.i.d. from the document-specific distribution over topics, while the tokens are drawn i.i.d. from the topics' distributions over words $\Phi = \{\boldsymbol{\phi}_1, \ldots, \boldsymbol{\phi}_T\}$:

$$P(\boldsymbol{z}^{(d)} \mid \boldsymbol{\theta}_d) = \prod_n \theta_{z_n^{(d)}|d} \quad \text{and} \quad P(\boldsymbol{w}^{(d)} \mid \boldsymbol{z}^{(d)}, \Phi) = \prod_n \phi_{w_n^{(d)}|z_n^{(d)}}. \qquad (2)$$

Dirichlet–multinomial conjugacy allows $\Theta$ and $\Phi$ to be marginalized out.

For real-world data, documents $\mathcal{W}$ are observed, while the corresponding topic assignments $\mathcal{Z}$ are unobserved. Variational methods [3, 16] and MCMC methods [7] are both effective at inferring the latent topic assignments $\mathcal{Z}$. Asuncion et al. [1] demonstrated that the choice of inference method has negligible effect on the probability of held-out documents or inferred topics. We use MCMC methods throughout this paper—specifically Gibbs sampling [5]—since the internal structure of hierarchical Dirichlet priors are typically inferred using a Gibbs sampling algorithm, which can be easily interleaved with Gibbs updates for $\mathcal{Z}$ given $\mathcal{W}$. The latter is accomplished by sequentially resampling each topic assignment $z_n^{(d)}$ from its conditional posterior given $\mathcal{W}$, $\alpha \boldsymbol{u}$, $\beta \boldsymbol{u}$ and $\mathcal{Z}_{\backslash d,n}$ (the current topic assignments for all tokens other than the token at position $n$ in document $d$):

$$P(z_n^{(d)} \mid \mathcal{W}, \mathcal{Z}_{\backslash d,n}, \alpha \boldsymbol{u}, \beta \boldsymbol{u}) \propto P(w_n^{(d)} \mid z_n^{(d)}, \mathcal{W}_{\backslash d,n}, \mathcal{Z}_{\backslash d,n}, \beta \boldsymbol{u}) \, P(z_n^{(d)} \mid \mathcal{Z}_{\backslash d,n}, \alpha \boldsymbol{u})$$

$$\propto \frac{N_{w_n^{(d)}|z_n^{(d)}}^{\backslash d,n} + \frac{\beta}{W}}{N_{z_n^{(d)}}^{\backslash d,n} + \beta} \frac{N_{z_n^{(d)}|d}^{\backslash d,n} + \frac{\alpha}{T}}{N_d - 1 + \alpha}, \qquad (3)$$

where sub- or super-script "$\backslash d, n$" denotes a quantity excluding data from position $n$ in document $d$.

## 3 Priors for LDA

The previous section outlined LDA as it is most commonly used—namely with symmetric Dirichlet priors over $\Theta$ and $\Phi$ with fixed concentration parameters $\alpha$ and $\beta$, respectively. The simplest way to vary this choice of prior for either $\Theta$ or $\Phi$ is to infer the relevant concentration parameter from data, either by computing a MAP estimate [1] or by using an MCMC algorithm such as slice sampling [13]. A broad Gamma distribution is an appropriate choice of prior for both $\alpha$ and $\beta$.

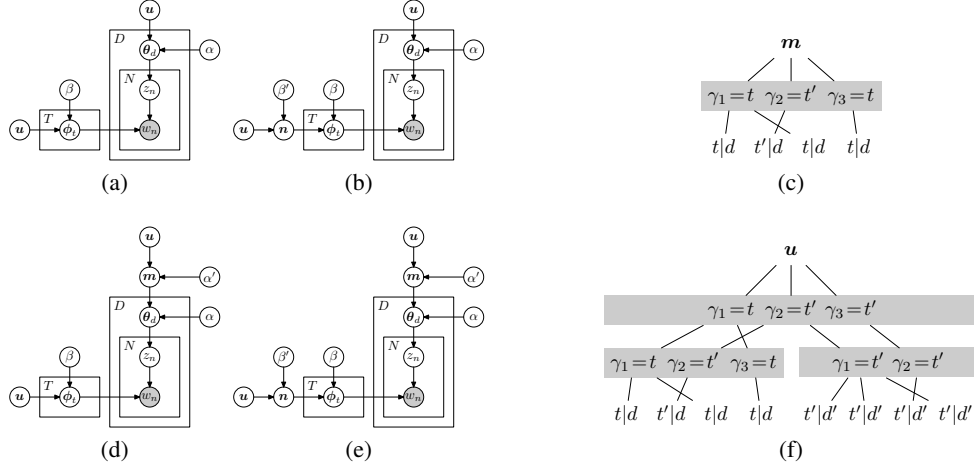

Figure 1: (a)-(e): LDA with (a) symmetric Dirichlet priors over $\Theta$ and $\Phi$, (b) a symmetric Dirichlet prior over $\Theta$ and an asymmetric Dirichlet prior over $\Phi$, (d) an asymmetric Dirichlet prior over $\Theta$ and a symmetric Dirichlet prior over $\Phi$, (e) asymmetric Dirichlet priors over $\Theta$ and $\Phi$. (c) Generating $\{z_n^{(d)}\}_{n=1}^4 = (t, t', t, t)$ from the asymmetric, predictive distribution for document $d$; (f) generating $\{z_n^{(d)}\}_{n=1}^4 = (t, t', t, t)$ and $\{z_n^{(d')}\}_{n=1}^4 = (t', t', t', t')$ from the asymmetric, hierarchical predictive distributions for documents $d$ and $d'$, respectively.

Alternatively, the uniform base measures in the Dirichlet priors over $\Theta$ and $\Phi$ can be replaced with nonuniform base measures $\boldsymbol{m}$ and $\boldsymbol{n}$, respectively. Throughout this section we use the prior over $\Theta$ as a running example, however the same construction and arguments also apply to the prior over $\Phi$. In section 3.1, we describe the effects on the document-specific conditional posterior distributions, or predictive distributions, of replacing $\boldsymbol{u}$ with a fixed asymmetric (i.e., nonuniform) base measure $\boldsymbol{m}$. In section 3.2, we then treat $\boldsymbol{m}$ as unknown, and take a fully Bayesian approach, giving $\boldsymbol{m}$ a Dirichlet prior (with a uniform base measure and concentration parameter $\alpha'$) and integrating it out.

## 3.1   Asymmetric Dirichlet Priors

If $\Theta$ is given an asymmetric Dirichlet prior with concentration parameter $\alpha$ and an known (nonuniform) base measure $\boldsymbol{m}$, the predictive probability of topic $t$ occurring in document $d$ given $\mathcal{Z}$ is

$$P(z_{N_d+1}^{(d)}=t \mid \mathcal{Z}, \alpha\boldsymbol{m}) = \int \mathrm{d}\boldsymbol{\theta}_d \, P(t \mid \boldsymbol{\theta}_d) \, P(\boldsymbol{\theta}_d \mid \mathcal{Z}, \alpha\boldsymbol{m}) = \frac{N_{t|d} + \alpha m_t}{N_d + \alpha}. \tag{4}$$

If topic $t$ does not occur in $\boldsymbol{z}^{(d)}$, then $N_{t|d}$ will be zero, and the probability of generating $z_{N_d+1}^{(d)}=t$ will be $m_t$. In other words, under an asymmetric prior, $N_{t|d}$ is smoothed with a topic-specific quantity $\alpha m_t$. Consequently, different topics can be a priori more or less probable in all documents.

One way of describing the process of generating from (4) is to say that generating a topic assignment $z_n^{(d)}$ is equivalent to setting the value of $z_n^{(d)}$ to the the value of some document-specific draw from $\boldsymbol{m}$. While this interpretation provides little benefit in the case of fixed $\boldsymbol{m}$, it is useful for describing the effects of marginalizing over $\boldsymbol{m}$ on the predictive distributions (see section 3.2). Figure 1c depicts the process of drawing $\{z_n^{(d)}\}_{n=1}^4$ using this interpretation. When drawing $z_1^{(d)}$, there are no existing document-specific draws from $\boldsymbol{m}$, so a new draw $\gamma_1$ must be generated, and $z_1^{(d)}$ assigned the value of this draw ($t$ in figure 1c). Next, $z_2^{(d)}$ is drawn by either selecting $\gamma_1$, with probability proportional to the number of topic assignments that have been previously "matched" to $\gamma_1$, or a new draw from $\boldsymbol{m}$, with probability proportional to $\alpha$. In figure 1c, a new draw is selected, so $\gamma_2$ is drawn from $\boldsymbol{m}$ and $z_2^{(d)}$ assigned its value, in this case $t'$. The next topic assignment is drawn in the same way: existing draws $\gamma_1$ and $\gamma_2$ are selected with probabilities proportional to the numbers of topic assignments to which they have previously been matched, while with probability proportional to $\alpha$, $z_3^{(d)}$ is matched to a new draw from $\boldsymbol{m}$. In figure 1c, $\gamma_1$ is selected and $z_3^{(d)}$ is assigned the value of $\gamma_1$. In general, the probability of a new topic assignment being assigned the value of an existing document-specific draw $\gamma_i$ from $\boldsymbol{m}$ is proportional to $N_d^{(i)}$, the number of topic assignments

previously matched to $\gamma_i$. The predictive probability of topic $t$ in document $d$ is therefore

$$P(z_{N_d+1}^{(d)} = t \mid \mathcal{Z}, \alpha\boldsymbol{m}) = \frac{\sum_{i=1}^{I} N_d^{(i)} \delta\left(\gamma_i - t\right) + \alpha m_t}{N_d + \alpha}, \tag{5}$$

where $I$ is the current number of draws from $\boldsymbol{m}$ for document $d$. Since every topic assignment is matched to a draw from $\boldsymbol{m}$, $\sum_{i=1}^{I} N_d^{(i)} \delta\left(\gamma_i - t\right) = N_{t|d}$. Consequently, (4) and (5) are equivalent.

## 3.2  Integrating out $m$

In practice, the base measure $\boldsymbol{m}$ is not fixed a priori and must therefore be treated as an unknown quantity. We take a fully Bayesian approach, and give $\boldsymbol{m}$ a symmetric Dirichlet prior with concentration parameter $\alpha'$ (as shown in figures 1d and 1e). This prior over $\boldsymbol{m}$ induces a *hierarchical* Dirichlet prior over $\Theta$. Furthermore, Dirichlet–multinomial conjugacy then allows $\boldsymbol{m}$ to be integrated out.

Giving $\boldsymbol{m}$ a symmetric Dirichlet prior and integrating it out has the effect of replacing $\boldsymbol{m}$ in (5) with a "global" Pólya conditional distribution, shared by the document-specific predictive distributions. Figure 1f depicts the process of drawing eight topic assignments—four for document $d$ and four for document $d'$. As before, when a topic assignment is drawn from the predictive distribution for document $d$, it is assigned the value of an existing (document-specific) internal draw $\gamma_i$ with probability proportional to the number of topic assignments previously matched to that draw, and to the value of a new draw $\gamma_{i'}$ with probability proportional to $\alpha$. However, since $\boldsymbol{m}$ has been integrated out, the new draw must be obtained from the "global" distribution. At this level, $\gamma_{i'}$ treated as if it were a topic assignment, and assigned the value of an existing *global* draw $\gamma_j$ with probability proportional to the number of document-level draws previously matched to $\gamma_j$, and to a new global draw, from $\boldsymbol{u}$, with probability proportional to $\alpha'$. Since the internal draws at the document level are treated as topic assignments the global level, there is a path from every topic assignment to $\boldsymbol{u}$, via the internal draws. The predictive probability of topic $t$ in document $d$ given $\mathcal{Z}$ is now

$$P(z_{N_d+1}^{(d)} = t \mid \mathcal{Z}, \alpha, \alpha'\boldsymbol{u}) = \int \mathrm{d}\boldsymbol{m}\, P(z_{N_d+1}^{(d)} = t \mid \mathcal{Z}, \alpha\boldsymbol{m})\, P(\boldsymbol{m} \mid \mathcal{Z}, \alpha'\boldsymbol{u})$$

$$= \frac{N_{t|d} + \alpha \dfrac{\hat{N}_t + \frac{\alpha'}{T}}{\sum_t \hat{N}_t + \alpha'}}{N_d + \alpha}, \tag{6}$$

where $I$ and $J$ are the current numbers of document-level and global internal draws, respectively, $N_{t|d} = \sum_{i=1}^{I} N_d^{(i)} \delta\left(\gamma_i - t\right)$ as before and $\hat{N}_t = \sum_{j=1}^{J} N^{(j)} \delta\left(\gamma_j - t\right)$. The quantity $N^{(j)}$ is the total number of document-level internal draws matched to global internal draw $\gamma_j$. Since some topic assignments will be matched to existing document-level draws, $\sum_d \delta\left(N_{t|d} > 0\right) \leq \hat{N}_t \leq N_t$, where $\sum_d \delta\left(N_{t|d} > 0\right)$ is the number of unique documents in $\mathcal{Z}$ in which topic $t$ occurs.

An important property of (6) is that if concentration parameter $\alpha'$ is large relative to $\sum_t \hat{N}_t$, then counts $\hat{N}_t$ and $\sum_t \hat{N}_t$ are effectively ignored. In other words, as $\alpha' \to \infty$ the hierarchical, asymmetric Dirichlet prior approaches a symmetric Dirichlet prior with concentration parameter $\alpha$.

For any given $\mathcal{Z}$ for real-world documents $\mathcal{W}$, the internal draws and the paths from $\mathcal{Z}$ to $\boldsymbol{u}$ are unknown. Only the value of each topic assignment is known, and hence $N_{t|d}$ for each topic $t$ and document $d$. In order to compute the conditional posterior distribution for each topic assignment (needed to resample $\mathcal{Z}$) it is necessary to infer $\hat{N}_t$ for each topic $t$. These values can be inferred by Gibbs sampling the paths from $\mathcal{Z}$ to $\boldsymbol{u}$ [4, 15]. Resampling the paths from $\mathcal{Z}$ to $\boldsymbol{u}$ can be interleaved with resampling $\mathcal{Z}$ itself. Removing $z_n^{(d)} = t$ from the model prior to resampling its value consists of decrementing $N_{t|d}$ and removing its current path to $\boldsymbol{u}$. Similarly, adding a newly sampled value $z_n^{(d)} = t'$ into the model consists of incrementing $N_{t'|d}$ and sampling a new path from $z_n^{(d)}$ to $\boldsymbol{u}$.

## 4  Comparing Priors for LDA

To investigate the effects of the priors over $\Theta$ and $\Phi$, we compared the four combinations of symmetric and asymmetric Dirichlets shown in figure 1: symmetric priors over both $\Theta$ and $\Phi$ (denoted

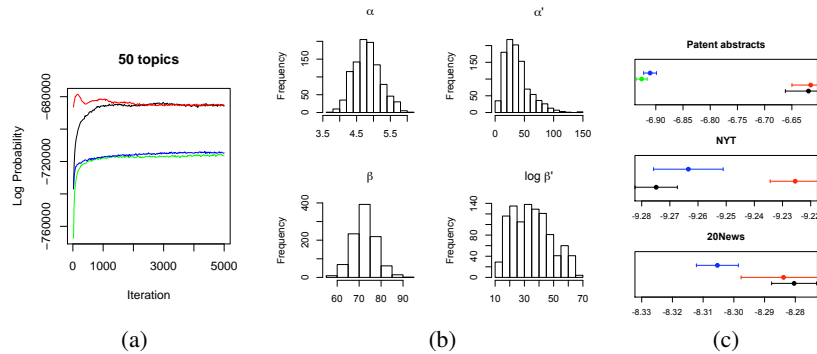

(a)                                    (b)                                    (c)

Figure 2: (a) $\log P(\mathcal{W}, \mathcal{Z} \mid \Omega)$ (patent abstracts) for SS, SA, AS and AS, computed every 20 iterations and averaged over 5 Gibbs sampling runs. AS (red) and AA (black) perform similarly and converge to higher values of $\log P(\mathcal{W}, \mathcal{Z} \mid \Omega)$ than SS (blue) and SA (green). (b) Histograms of 4000 (iterations 1000-5000) concentration parameter values for AA (patent abstracts). Note the log scale for $\beta'$: the prior over $\Phi$ approaches a symmetric Dirichlet, making AA equivalent to AS. (c) $\log P(\mathcal{W}, \mathcal{Z} \mid \Omega)$ for all three data sets at $T = 50$. AS is consistently better than SS. SA is poor (not shown). AA is capable of matching AS, but does not always.

| Data set | $D$ | $\bar{N}_d$ | $N$ | $W$ | Stop |
|---|---|---|---|---|---|
| Patent abstracts | 1016 | 101.87 | 103499 | 6068 | yes |
| 20 Newsgroups | 540 | 148.17 | 80012 | 14492 | no |
| NYT articles | 1768 | 270.06 | 477465 | 41961 | no |

Table 1: Data set statistics. $D$ is the number of documents, $\bar{N}_d$ is the mean document length, $N$ is the number of tokens, $W$ is the vocabulary size. "Stop" indicates whether stop words were present (yes) or not (no).

SS), a symmetric prior over $\Theta$ and an asymmetric prior over $\Phi$ (denoted SA), an asymmetric prior over $\Theta$ and a symmetric prior over $\Phi$ (denoted AS), and asymmetric priors over both $\Theta$ and $\Phi$ (denoted AA). Each combination was used to model three collections of documents: patent abstracts about carbon nanotechnology, New York Times articles, and 20 Newsgroups postings. Due to the computationally intensive nature of the fully Bayesian inference procedure, only a subset of each collection was used (see table 1). In order to stress each combination of priors with respect to skewed distributions over word frequencies, stop words were not removed from the patent abstracts.

The four models (SS, SA, AS, AA) were implemented in Java, with integrated-out base measures, where appropriate. Each model was run with $T \in \{25, 50, 75, 100\}$ for five runs of 5000 Gibbs sampling iterations, using different random initializations. The concentration parameters for each model (denoted by $\Omega$) were given broad Gamma priors and inferred using slice sampling [13]. During inference, $\log P(\mathcal{W}, \mathcal{Z} \mid \Omega)$ was recorded every twenty iterations. These values, averaged over the five runs for $T = 50$, are shown in figure 2a. (Results for other values of $T$ are similar.) There are two distinct patterns: models with an asymmetric prior over $\Theta$ (AS and AA; red and black, respectively) perform very similarly, while models with a symmetric prior over $\Theta$ (SS and SA; blue and green, respectively) also perform similarly, with significantly worse performance than AS and AA. Results for all three data sets are summarized in figure 2c, with the log probability divided by the number of tokens in the collection. SA performs extremely poorly on NYT and 20 Newsgroups, and is not therefore shown. AS consistently achieves better likelihood than SS. The fully asymmetric model, AA, is inconsistent, matching AS on the patents and 20 Newsgroups but doing poorly on NYT. This is most likely due to the fact that although AA can match AS, it has many more degrees of freedom and therefore a much larger space of possibilities to explore.

We also calculated the probability of held-out documents using the "left-to-right" evaluation method described by Wallach et al. [17]. These results are shown in figure 3a, and exhibit a similar pattern to the results in figure 2a—the best-performing models are those with an asymmetric priors over $\Theta$.

We can gain intuition about the similarity between AS and AA by examining the values of the sampled concentration parameters. As explained in section 3.2, as $\alpha'$ or $\beta'$ grows large relative to $\sum_t \hat{N}_t$ or $\sum_w \hat{N}_w$, an asymmetric Dirichlet prior approaches a symmetric Dirichlet with concentration parameter $\alpha$ or $\beta$. Histograms of 4000 concentration parameter values (from iterations 1000-4000) from the five Gibbs runs of AA with $T = 50$ are shown in figure 2b. The values for $\alpha$, $\alpha'$ and $\beta$

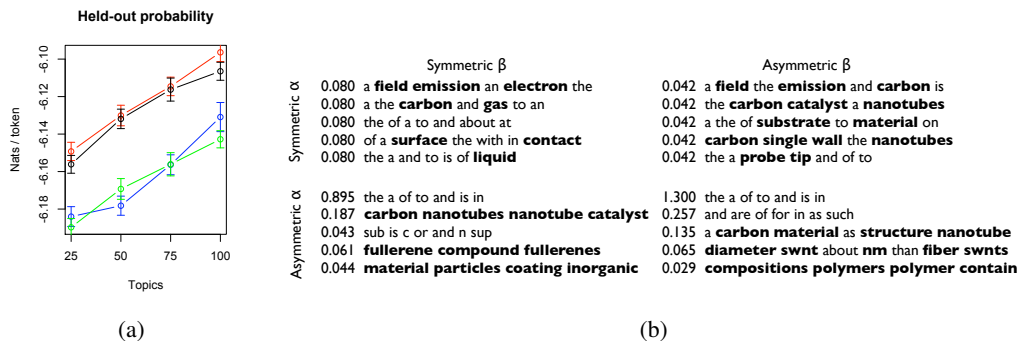

Figure 3: (a) Log probability of held-out documents (patent abstracts). These results mirror those in figure 2a. AS (red) and AA (black) again perform similarly, while SS (blue) and SA (green) are also similar, but exhibit much worse performance. (b) $\alpha m_t$ values and the most probable words for topics obtained with $T = 50$. For each model, topics were ranked according to usage and the topics at ranks 1, 5, 10, 20 and 30 are shown. AS and AA are robust to skewed word frequency distributions and tend to sequester stop words in their own topics.

are all relatively small, while the values for $\beta'$ are extremely large, with a median around $\exp 30$. In other words, given the values of $\beta'$, the prior over $\Phi$ is *effectively* a symmetric prior over $\Phi$ with concentration parameter $\beta$. These results demonstrate that even when the model can use an asymmetric prior over $\Phi$, a symmetric prior gives better performance. We therefore advocate using model AS.

It is worth noting the robustness of AS to stop words. Unlike SS and SA, AS effectively sequesters stop words in a small number of more frequently used topics. The remaining topics are relatively unaffected by stop words. Creating corpus-specific stop word lists is seen as an unpleasant but necessary chore in topic modeling. Also, for many specialized corpora, once standard stop words have been removed, there are still other words that occur with very high probability, such as "model," "data," and "results" in machine learning literature, but are not technically stop words. If LDA cannot handle such words in an appropriate fashion then they must be treated as stop words and removed, despite the fact that they play meaningful semantic roles. The robustness of AS to stop words has implications for HMM-LDA [8] which models stop words using a hidden Markov model and "content" words using LDA, at considerable computational cost. AS achieves the same robustness to stop words much more efficiently. Although there is empirical evidence that topic models that use asymmetric Dirichlet priors with optimized hyperparameters, such as Pachinko allocation [10] and Wallach's topic-based language model [18], are robust to the presence of extremely common words, these studies did not establish whether the robustness was a function of a more complicated model structure or if careful consideration of hyperparameters alone was sufficient. We demonstrate that AS is capable of learning meaningful topics even with no stop word removal. For efficiency, we do not necessarily advocate doing away with stop word lists entirely, but we argue that using an asymmetric prior over $\Theta$ allows practitioners to use a standard, conservative list of determiners, prepositions and conjunctions that is applicable to any document collection in a given language, rather than hand-curated corpus-specific lists that risk removing common but meaningful terms.

# 5 Efficiency: Optimizing rather than Integrating Out

Inference in the full Bayesian formulation of AS is expensive because of the additional complexity in sampling the paths from $\mathcal{Z}$ to $\boldsymbol{u}$ and maintaining hierarchical data structures. It is possible to retain the theoretical and practical advantages of using AS without sacrificing the advantages of simple, efficient models by directly optimizing $\boldsymbol{m}$, rather than integrating it out. The concentration parameters $\alpha$ and $\beta$ may also be optimized (along with $\boldsymbol{m}$ for $\alpha$ and by itself for $\beta$). In this section, we therefore compare the fully Bayesian version of AS with optimized AS, using SS as a baseline.

Wallach [19] compared several methods for jointly the maximum likelihood concentration parameter and asymmetric base measure for a Dirichlet–multinomial model. We use the most efficient of these methods. The advantage of optimizing $\boldsymbol{m}$ is considerable: although it is likely that further optimizations would reduce the difference, 5000 Gibbs sampling iterations (including sampling $\alpha$,

| | Patents | NYT | 20 NG | | | 25 | 50 | 75 | 100 |
|---|---|---|---|---|---|---|---|---|---|
| ASO | -6.65 ± 0.04 | -9.24 ± 0.01 | -8.27 ± 0.01 | | ASO | -6.18 | **-6.12** | -6.12 | **-6.08** |
| AS | -6.62 ± 0.03 | -9.23 ± 0.01 | -8.28 ± 0.01 | | AS | **-6.15** | -6.13 | **-6.11** | -6.10 |
| SS | -6.91 ± 0.01 | -9.26 ± 0.01 | -8.31 ± 0.01 | | SS | -6.18 | -6.18 | -6.16 | -6.13 |

Table 2: $\log P(\mathcal{W}, \mathcal{Z} \,|\, \Omega) \,/\, N$ for $T = 50$ (left) and $\log P(\mathcal{W}^{\text{test}} \,|\, \mathcal{W}, \mathcal{Z}, \Omega) \,/\, N^{\text{test}}$ for varying values of $T$ (right) for the patent abstracts. AS and ASO (optimized hyperparameters) consistently outperform SS except for ASO with $T = 25$. Differences between AS and ASO are inconsistent and within standard deviations.

| | ASO | AS | SS | | | ASO | AS | SS |
|---|---|---|---|---|---|---|---|---|
| ASO | 4.37 ± 0.08 | 4.34 ± 0.09 | 5.43 ± 0.05 | | ASO | 3.36 ± 0.03 | 3.43 ± 0.05 | 3.50 ± 0.07 |
| AS | — | 4.18 ± 0.09 | 5.39 ± 0.06 | | AS | — | 3.36 ± 0.02 | 3.56 ± 0.07 |
| SS | — | — | 5.93 ± 0.03 | | SS | — | — | 3.49 ± 0.04 |

Table 3: Average VI distances between multiple runs of each model with $T = 50$ on (left) patent abstracts and (right) 20 newsgroups. ASO partitions are approximately as similar to AS partitions as they are to other ASO partitions. ASO and AS partitions are both are further from SS partitions, which tend to be more dispersed.

$\alpha'$ and $\beta$) for the patent abstracts using fully Bayesian AS with $T = 25$ took over four hours, while 5000 Gibbs sampling iterations (including hyperparameter optimization) took under 30 minutes.

In order to establish that optimizing $m$ is a good approximation to integrating it out, we computed $\log P(\mathcal{W}, \mathcal{Z} \,|\, \Omega)$ and the log probability of held-out documents for fully Bayesian AS, optimized AS (denoted ASO) and as a baseline SS (see table 2). AS and ASO consistently outperformed SS, except for ASO when $T = 25$. Since twenty-five is a very small number of topics, this is not a cause for concern. Differences between AS and ASO are inconsistent and within standard deviations. From a point of view of log probabilities, ASO therefore provides a good approximation to AS.

We can also compare topic assignments. Any set of topic assignments can be thought of as partition of the corresponding tokens into $T$ topics. In order to measure similarity between two sets of topic assignments $\mathcal{Z}$ and $\mathcal{Z}'$ for $\mathcal{W}$, we can compute the distance between these partitions using *variation of information* (VI) [11, 6] (see suppl. mat. for a definition of VI for topic models). VI has several attractive properties: it is a proper distance metric, it is invariant to permutations of the topic labels, and it can be computed in $\mathcal{O}\,(N + TT')$ time, i.e., time that is linear in the number of tokens and the product of the numbers of topics in $\mathcal{Z}$ and $\mathcal{Z}'$. For each model (AS, ASO and SS), we calculated the average VI distance between all 10 unique pairs of topic assignments from the 5 Gibbs runs for that model, giving a measure of within-model consistency. We also calculated the between-model VI distance for each pair of models, averaged over all 25 unique pairs of topic assignments for that pair. Table 3 indicates that ASO partitions are approximately as similar to AS partitions as they are to other ASO partitions. ASO and AS partitions are both further away from SS partitions, which tend to be more dispersed. These results confirm that ASO is indeed a good approximation to AS.

## 6    Effect on Selecting the Number of Topics

Selecting the number of topics $T$ is one of the most problematic modeling choices in finite topic modeling. Not only is there no clear method for choosing $T$ (other than evaluating the probability of held-out data for various values of $T$), but degree to which LDA is robust to a poor setting of $T$ is not well-understood. Although nonparametric models provide an alternative, they lose the substantial computational efficiency advantages of finite models. We explore whether the combination of priors advocated in the previous sections (model AS) can improve the stability of LDA to different values of $T$, while retaining the static memory management and simple inference algorithms of finite models.

Ideally, if LDA has sufficient topics to model $\mathcal{W}$ well, the assignments of tokens to topics should be relatively invariant to an increase in $T$—i.e., the additional topics should be seldom used. For example, if ten topics is sufficient to accurately model the data, then increasing the number of topics to twenty shouldn't significantly affect inferred topic assignments. If this is the case, then using large $T$ should not have a significant impact on either $\mathcal{Z}$ or the speed of inference, especially as recently-introduced sparse sampling methods allow models with large $T$ to be trained efficiently [20]. Figure 4a shows the average VI distance between topic assignments (for the patent abstracts) inferred by models with $T = 25$ and models with $T \in \{50, 75, 100\}$. AS and AA, the bottom two lines, are

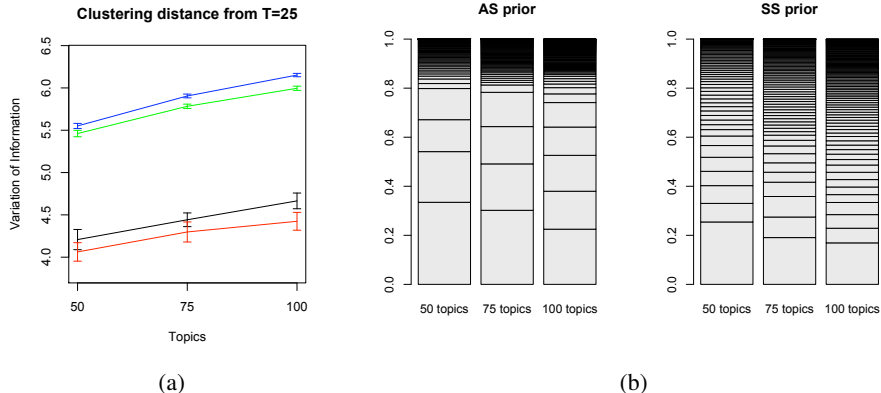

(a)                                  (b)

Figure 4: (a) Topic consistency measured by average VI distance from models with $T = 25$. As $T$ increases, AS (red) and AA (black) produce $\mathcal{Z}$s that stay significantly closer to those obtained with $T = 25$ than SA (green) and SS (blue). (b) Assignments of tokens (patent abstracts) allocated to the largest topic in a 25 topic model, as $T$ increases. For AS, the topic is relatively intact, even at $T = 100$: 80% of tokens assigned to the topic at $T = 25$ are assigned to seven topics. For SS, the topic has been subdivided across many more topics.

much more stable (smaller average VI distances) than SS and SA at 50 topics and remain so as $T$ increases: even at 100 topics, AS has a smaller VI distance to a 25 topic model than SS at 50 topics. Figure 4b provides intuition for this difference: for AS, the tokens assigned to the largest topic at $T = 25$ remain within a small number of topics as $T$ is increased, while for SS, topic usage is more uniform and increasing $T$ causes the tokens to be divided among many more topics. These results suggest that for AS, new topics effectively "nibble away" at existing topics, rather than splitting them more uniformly. We therefore argue that the risk of using too many topics is lower than the risk of using too few, and that practitioners should be comfortable using larger values of $T$.

# 7 Discussion

The previous sections demonstrated that AS results in the best performance over AA, SA and SS, measured in several ways. However, it is worth examining *why* this combination of priors results in superior performance. The primary assumption underlying topic modeling is that a topic should capture semantically-related word co-occurrences. Topics must also be distinct in order to convey information: knowing only a few co-occurring words should be sufficient to resolve semantic ambiguities. A priori, we therefore do not expect that a particular topic's distribution over words will be like that of any other topic. An asymmetric prior over $\Phi$ is therefore a bad idea: the base measure will reflect corpus-wide word usage statistics, and a priori, all topics will exhibit those statistics too. A symmetric prior over $\Phi$ only makes a prior statement (determined by the concentration parameter $\beta$) about whether topics will have more sparse or more uniform distributions over words, so the topics are free to be as distinct and specialized as is necessary. However, it is still necessary to account for power-law word usage. A natural way of doing this is to expect that certain groups of words will occur more frequently than others *in every document* in a given corpus. For example, the words "model," "data," and "algorithm" are likely to appear in every paper published in a machine learning conference. These assumptions lead naturally to the combination of priors that we have empirically identified as superior: an asymmetric Dirichlet prior over $\Theta$ that serves to share commonalities across documents and a symmetric Dirichlet prior over $\Phi$ that serves to avoid conflicts between topics. Since these priors can be implemented using efficient algorithms that add negligible cost beyond standard inference techniques, we recommend them as a new standard for LDA.

# 8 Acknowledgments

This work was supported in part by the Center for Intelligent Information Retrieval, in part by CIA, NSA and NSF under NSF grant number IIS-0326249, and in part by subcontract number B582467 from Lawrence Livermore National Security, LLC under prime contract number DE-AC52-07NA27344 from DOE/NNSA. Any opinions, findings and conclusions or recommendations expressed in this material are the authors' and do not necessarily reflect those of the sponsor.

## Footnotes

[1]http://rexa.info/

# References

[1] A. Asuncion, M. Welling, P. Smyth, and Y. W. Teh. On smoothing and inference for topic models. In *Proceedings of the 25th Conference on Uncertainty in Artificial Intelligence*, 2009.

[2] D. Blei and J. Lafferty. A correlated topic model of Science. *Annals of Applied Statistics*, 1(1):17–35, 2007.

[3] D. M. Blei, A. Y. Ng, and M. I. Jordan. Latent Dirichlet allocation. *Journal of Machine Learning Research*, 3:993–1022, January 2003.

[4] P. J. Cowans. *Probabilistic Document Modelling*. PhD thesis, University of Cambridge, 2006.

[5] S. Geman and D. Geman. Stochastic relaxation, Gibbs distributions, and the Bayesian restoration of images. *IEEE Transaction on Pattern Analysis and Machine Intelligence 6*, pages 721–741, 1984.

[6] S. Goldwater and T. L. Griffiths. A fully Bayesian approach to unsupervised part-of-speech tagging. In *Association for Computational Linguistics*, 2007.

[7] T. L. Griffiths and M. Steyvers. Finding scientific topics. *Proceedings of the National Academy of Sciences*, 101(suppl. 1):5228–5235, 2004.

[8] T. L. Griffiths, M. Steyvers, D. M. Blei, and J. B. Tenenbaum. Integrating topics and syntax. In L. K. Saul, Y. Weiss, and L. Bottou, editors, *Advances in Neural Information Processing Systems 17*, pages 536–544. The MIT Press, 2005.

[9] D. Hall, D. Jurafsky, and C. D. Manning. Studying the history of ideas using topic models. In *Proceedings of EMNLP 2008*, pages 363–371.

[10] W. Li and A. McCallum. Mixtures of hierarchical topics with pachinko allocation. In *Proceedings of the 24th International Conference on Machine learning*, pages 633–640, 2007.

[11] M. Meilă. Comparing clusterings by the variation of information. In *Conference on Learning Theory*, 2003.

[12] D. Mimno and A. McCallum. Organizing the OCA: Learning faceted subjects from a library of digital books. In *Proceedings of the 7th ACM/IEEE joint conference on Digital libraries*, pages 376–385, Vancouver, BC, Canada, 2007.

[13] R. M. Neal. Slice sampling. *Annals of Statistics*, 31:705–767, 2003.

[14] D. Newman, C. Chemudugunta, P. Smyth, and M. Steyvers. Analyzing entities and topics in news articles using statistical topic models. In *Intelligence and Security Informatics*, Lecture Notes in Computer Science. 2006.

[15] Y. W. Teh, M. I. Jordan, M. J. Beal, and D. M. Blei. Hierarchical Dirichlet processes. *Journal of the American Statistical Association*, 101:1566–1581, 2006.

[16] Y. W. Teh, D. Newman, and M. Welling. A collapsed variational Bayesian inference algorithm for latent Dirichlet allocation. In *Advances in Neural Information Processing Systems 18*, 2006.

[17] H. Wallach, I. Murray, R. Salakhutdinov, and D. Mimno. Evaluation methods for topic models. In *Proceedings of the 26th Interational Conference on Machine Learning*, 2009.

[18] H. M. Wallach. Topic modeling: Beyond bag-of-words. In *Proceedings of the 23rd International Conference on Machine Learning*, pages 977–984, Pittsburgh, Pennsylvania, 2006.

[19] H. M. Wallach. *Structured Topic Models for Language*. Ph.D. thesis, University of Cambridge, 2008.

[20] L. Yao, D. Mimno, and A. McCallum. Efficient methods for topic model inference on streaming document collections. In *Proceedings of KDD 2009*, 2009.

